# Hidden Markov Models for Human Genes

**Pierre Baldi** *
Jet Propulsion Laboratory
California Institute of Technology
Pasadena, CA 91109

**Søren Brunak**
Center for Biological Sequence Analysis
The Technical University of Denmark
DK-2800 Lyngby, Denmark

**Yves Chauvin** †
Net-ID, Inc.
601 Minnesota
San Francisco, CA 94107

**Jacob Engelbrecht**
Center for Biological Sequence Analysis
The Technical University of Denmark
DK-2800 Lyngby, Denmark

**Anders Krogh**
Electronics Institute
The Technical University of Denmark
DK-2800 Lyngby, Denmark

## Abstract

Human genes are not continuous but rather consist of short coding regions (exons) interspersed with highly variable non-coding regions (introns). We apply HMMs to the problem of modeling exons, introns and detecting splice sites in the human genome. Our most interesting result so far is the detection of particular oscillatory patterns, with a minimal period of roughly 10 nucleotides, that seem to be characteristic of exon regions and may have significant biological implications.

*and Division of Biology, California Institute of Technology.
†and Department of Psychology, Stanford University.

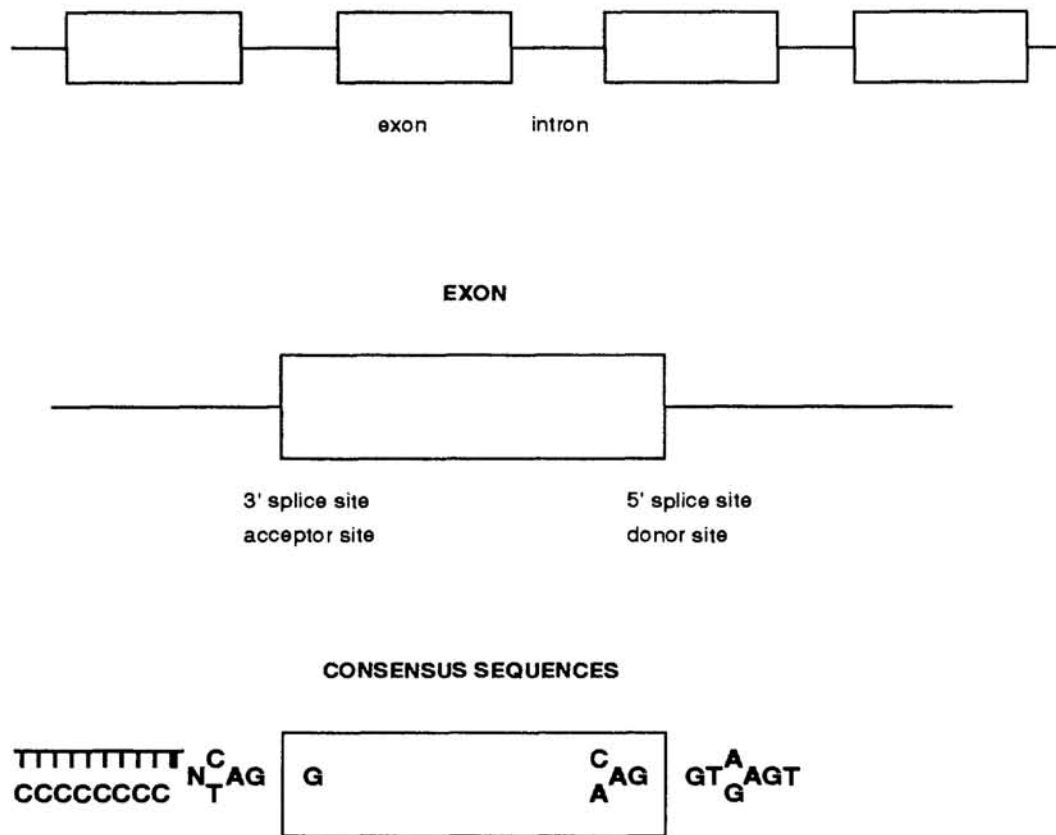

Figure 1: Structure of eukaryotic genes (not to scale: introns are typically much longer than exons).

# 1   INTRODUCTION

The genes of higher organisms are not continuous. Rather, they consist of relatively short coding regions called exons interspersed with non-coding regions of highly variable length called introns (Fig. 1). A complete gene may comprise as many as fifty exons. Very often, exons encode discrete functional or structural units of proteins. Prior to the translation of genes into proteins, a complex set of biochemical mechanisms is responsible for the precise cutting of genes at the splice junctions, i.e. the boundaries between introns and exons, and the subsequent removal and ligation which results in the production of mature messenger RNA. The translation machinery of the cell operates directly onto the mRNA, converting a primary sequence of nucleotides into the corresponding primary sequence of amino acids, according to the rules of the genetic code. The genetic code converts every three contiguous nucleotides, or codons, into one of the twenty amino acids (or into a stop signal). Therefore the splicing process must be exceedingly precise since a shift of only one base pair completely upsets the codon reading frame for translation. Many details of the splicing process are not known; in particular it is not clear how acceptor sites (i.e. intron/exon boundaries) and donor sites (i.e. exon/intron boundaries) are recognized with extremely high accuracy. Both acceptor and donor sites are signaled by the existence of consensus sequences, i.e. short sequences of nucleotides which are highly conserved across genes and, to some extent, across species. For instance,

most introns start with `GT` and terminate with `AG` and additional patterns can be detected in the proximity of the splice sites. The main problem with consensus sequences, in addition to their variability, is that by themselves they are insufficient for reliable splice site detection. Indeed, whereas exons are relatively short with an average length around 150 nucleotides, introns are often much longer, with several thousand of seemingly random nucleotides. Therefore numerous false positive consensus signals are bound to occur inside the introns. The `GT` dinucleotide constitutes roughly 5% of the dinucleotides in human DNA, but only a very small percentage of these belongs to the splicing donor category, in the order of 1.5%. The dinucleotide `AG` constitutes roughly 7.5% of all the dinucleotides and only around 1% of these function as splicing acceptor sites. In addition to consensus sequences at the splice sites, there seem to exist a number of other weak signals (Senapathy (1989), Brunak et al. (1992)) embedded in the 100 intron nucleotides upstream and downstream of an exon. Partial experimental evidence seems also to suggest that the recognition of the acceptor and donor boundaries of an exon may be a concerted process.

In connection with the current exponential growth of available DNA sequences and the human genome project, it has become essential to be able to algorithmically detect the boundaries between exons and introns and to parse entire genes. Unfortunately, current available methods are far from performing at the level of accuracy required for a systematic parsing of the entire human genome. Most likely, gene parsing requires the statistical integration of several weak signals, some of which are poorly known, over length scales of a few hundred nucleotides. Furthermore, initial and terminal exons, lacking one of the splice sites, need to be treated separately.

## 2    HMMs FOR BIOLOGICAL PRIMARY SEQUENCES

The parsing problem has been tackled with classical statistical methods and more recently using neural networks (Lapedes (1988), Brunak (1991)), with encouraging results. Conventional neural networks, however, do not seem ideally suited to handle the sort of elastic deformations introduced by evolutionary tinkering in genetic sequences. Another trend in recent years, has been the casting of DNA and protein sequences problems in terms of formal languages using context free grammars, automata and Hidden Markov Models (HMMs). The combination of machine learning techniques which can take advantage of abundant data together with new flexible representations appears particularly promising. HMMs in particular have been used to model protein families and address a number of task such as multiple alignments, classification and data base searches (Baldi et al. (1993) and (1994); Haussler et al. (1993); Krogh et al. (1994a); and references therein). It is the success obtained with this method on protein sequences and the ease with which it can handle insertions and deletions that naturally suggests its application to the parsing problem.

In Krogh et al. (1994b), HMMs are applied to the problem of detecting coding/non-coding regions in bacterial DNA (E. coli), which is characterized by the absence of true introns (like other prokaryotes). Their approach leads to a HMM that integrates both genic and intergenic regions, and can be used to locate genes fairly reliably. A similar approach for human DNA, that is not based on HMMs, but uses dynamic programming and neural networks to combine various gene finding techniques, is described in Snyder and Stormo (1993). In this paper we take a

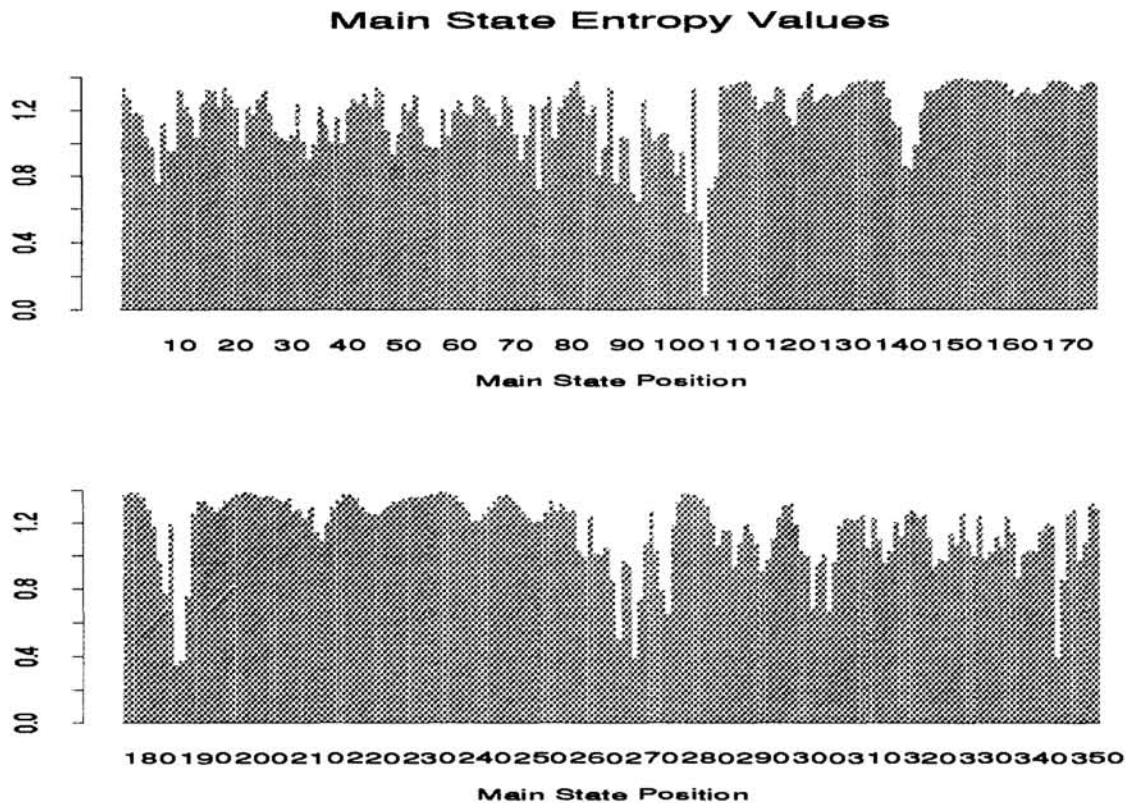

Figure 2: Entropy of emission distribution of main states.

first step towards parsing the human genome with HMMs by modeling exons (and flanking intron regions).

As in the applications of HMMs to speech or protein modeling, we use left-right architectures to model exon regions, intron regions or their boundaries. The architectures typically consist of a backbone of main states flanked by a sequence of delete states and a sequence of insert states, with the proper interconnections (see Baldi et al. (1994) and Krogh et al. (1994) for more details and Fig. 4 below). The data base used in the experiments to be described consists of roughly 2,000 human internal exons, with the corresponding adjacent introns, extracted from release 78 of the GenBank data base. It is essential to remark that, unlike in the previous experiments on protein families, the exons in the data base are not directly related by evolution. As a result, insertions and deletions in the model should be interpreted in terms of formal operations on the strings rather than evolutionary events.

## 3   EXPERIMENTS AND RESULTS

A number of different HMM training experiments have been carried using different classes of sequences including exons only, flanked exons (with 50 or 100 nucleotides on each side), introns only, flanked acceptor and flanked donor sites (with 100 nucleotides on each side) and slightly different architectures and learning algorithms. Only a few relevant examples will be given here.

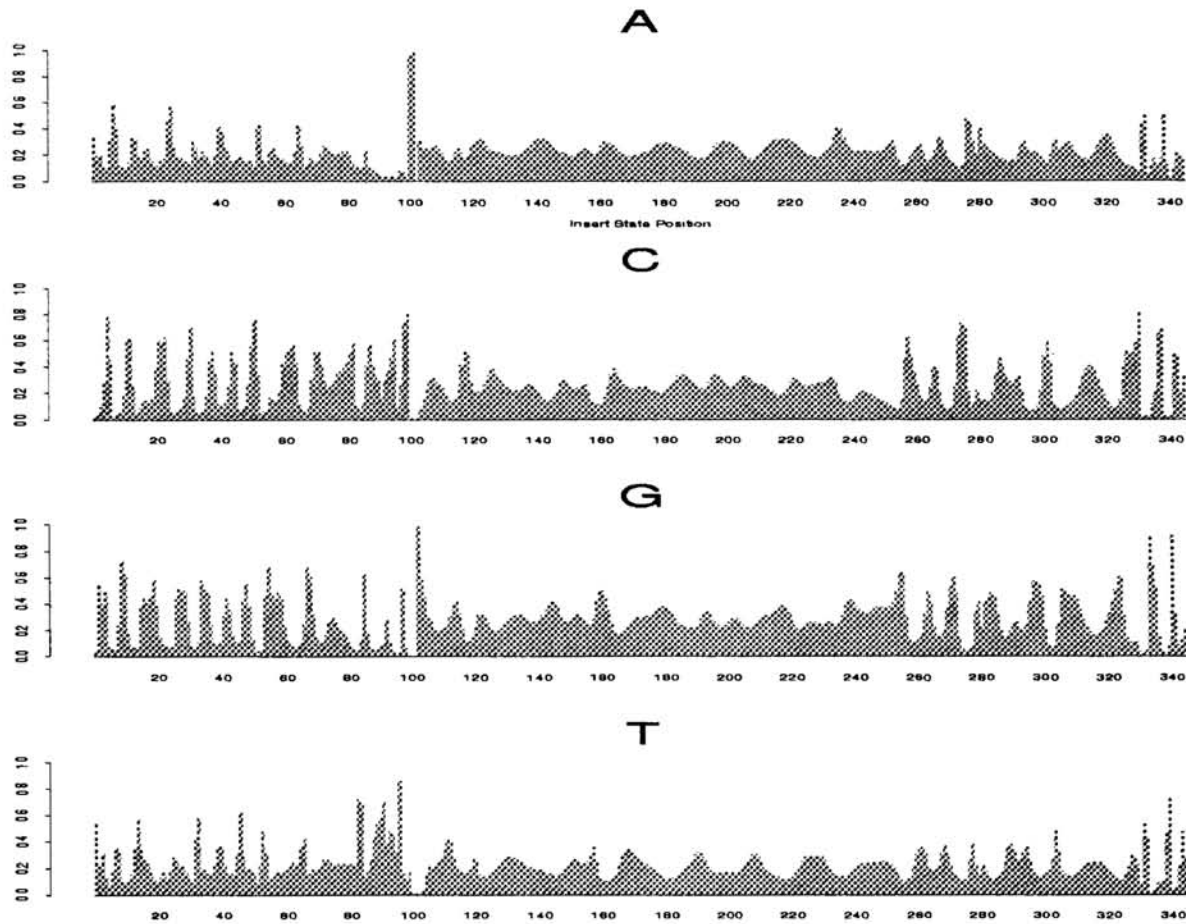

Figure 3: Emission distribution from main states.

In an early experiment, we trained a model of length 350 using 500 flanked exons, with 100 nucleotides on each side, using gradient descent on the negative log-likelihood (Baldi and Chauvin (1994)). The exons themselves had variable lengths between 50 and 300. The entropy plot (Fig. 2), after 7 gradient descent training cycles, reveals that the HMM has learned the acceptor site quite well but appears to have some difficulties with the donor site. One possible contributing factor is the high variability of the length of the training exons: the model seems to learn two donor sites, one for short exons and one for the other exons. The most striking pattern, however, is the greater smoothness of the entropy in the exon region. In the exon region, the entropy profile is weakly oscillatory, with a period of about 20 base pairs. Discrimination and t-tests conducted on this model show that it is definitely capable of discriminating exon regions, but the confidence level is not sufficient yet to reliably search entire genomes.

A slightly different model was subsequently trained using again 500 flanked exons, with the length of the exons between 100 and 200 only. The probability of emitting each one of the four nucleotides, across the main states of the model, are plotted in Fig. 3, after the sixth gradient descent training cycle. Again the donor site seems harder to learn than the acceptor site. Even more striking are the clear

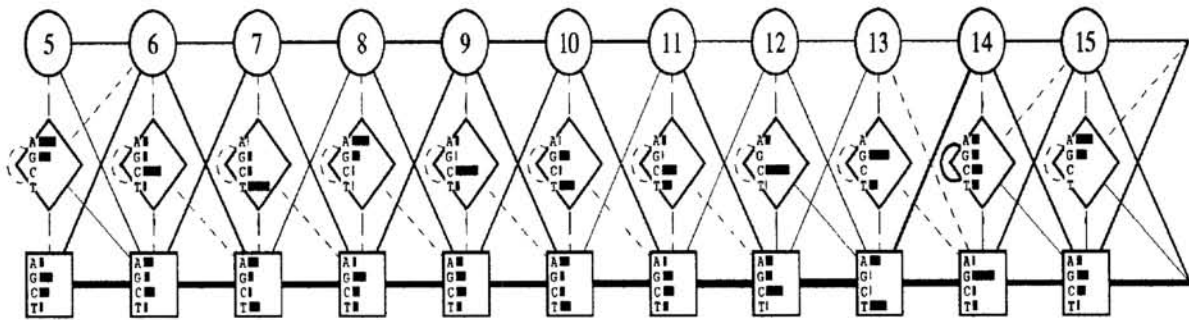

Figure 4: The repeated segment of the tied model. Note that position 15 is identical to position 5.

oscillatory patterns present in the exon region, characterized by a minimal period of 10 nucleotides, with A and G in phase and C and T in anti-phase.

The fact that the acceptor site is easier to learn could result from the fact that exons in the training sequences are always flanked by *exactly* 100 nucleotides upstream. To test this hypothesis, we trained a similar model using the same sequences but in *reverse* order. Surprisingly, the model still learns the acceptor site (which is now downstream from the donor site) much better than the donor site. The oscillatory pattern in the reversed exon region is still present. The oscillations we observe could also be an artifact of the method: for instance, when presented with random training sequences, oscillatory HMM solutions could appear naturally as local optima of the training procedure. To test this hypothesis, we trained a model using random sequences of similar average composition as the exons and found no distinct oscillatory patterns. We also checked that our data base of exons does not correspond prevalently to $\alpha$-helical domains of proteins.

To further test our findings, we trained a tied exon model with a hard-wired periodicity of 10. The tied model consists of 14 identical segments of length 10 and 5 additional positions in the beginning and end of the model, making a total length of 150. During training the segments are kept identical by *tying* of the parameters, i.e. the parameters are constrained to be exactly the same throughout learning, as in the weight sharing procedure for neural networks. The model was trained on 800 exon sequences of length between 100 and 200, and it was tested on 262 different sequences. The parameters of the repeated segment, after training, are shown in Fig. 4. Emission probabilities are represented by horizontal bars of corresponding proportional length. There is a lot of structure in this segment. The most prominent feature is the regular expression [^T][AT]G at position 12–14. (The regular expression means "anything but T followed by A or T followed by G".) The same pattern was often found at positions with very low entropy in the "standard models" described above. In order to test the significance, the tied model was compared to a standard model of the same length. The average negative log-likelihood (NNL) they both assign to the exon sequences and to random sequences of similar composition, as well as their number of parameters are shown in the table below.

| Model Scores | NLL training | NLL testing | # parameters |
|---|---|---|---|
| Standard model with random seqs | 203.2 | 200.3 | 2550 |
| Standard model with real seqs | 198.8 | 196.4 | 2550 |
| Tied model with real seqs | 198.6 | 195.6 | 340 |

The tied model achieves a level of performance comparable to the standard model but with significantly less free parameters, and therefore a period of 10 in the exons seems to be a strong hypothesis. Note that the period of the pattern is not strictly 10, and we found almost equally good models with a built-in period of 9 or 11.

The type of left-to-right architecture we have used is not the ideal model of an exon, because of the large length variations. It would be desirable to have a model with a loop structure such that the segment can be entered as many times as necessary for any given exon (see Krogh et al. (1994b) for a loop structure used for E. coli DNA). This is one of the future lines of research.

## 4    CONCLUSION

In summary, we are applying HMMs and related methods to the problems of exon/intron modeling and human genome parsing. Our preliminary results show that acceptor sites are intrinsically easier to learn than donor sites and that very simple HMM models alone are not sufficient for reliable genome parsing. Most importantly, interesting statistical 10 base oscillatory patterns have been detected in the exon regions. If confirmed, these patterns could have significant biological and algorithmic implications. These patterns could be related to the superimposition of several simultaneous codes (such as triplet code and frame code), and/or to the way DNA is wrapped around histone molecules (Beckmann and Trifonov (1991)). Presently, we are investigating their relationship to reading frame effects by training several HMM models using a data base of exons with the same reading frame.

### References

Beckmann, J.S. and Trifonov, E.N. (1991) Splice Junctions Follow a 205-base Ladder. PNAS USA, **88**, 2380-2383.

Baldi, P., Chauvin, Y., Hunkapiller, T. and McClure, M. A. (1994) Hidden Markov Models of Biological Primary Sequence Information. PNAS USA, 91, 3, 1059-1063.

Baldi, P., Chauvin, Y., Hunkapiller, T. and McClure, M. A. (1993) Hidden Markov Models in Molecular Biology: New Algorithms and Applications. Advances in Neural Information Processing Systems 5, Morgan Kaufmann, 747-754.

Baldi, P. and Chauvin, Y. (1994) Smooth On-Line Learning Algorithms for Hidden Markov Models. Neural Computation, **6**, 2, 305-316.

Brunak, S., Engelbrecht, J. and Knudsen, S. (1991) Prediction of Human mRNA Donor and Acceptor Sites from the DNA Sequence. Journal of Molecular Biology, **220**, 49-65.

Engelbrecht, J., Knudsen, S. and Brunak S., (1992) G/C rich tract in 5' end of human introns, Journal of Molecular Biology, **227**, 108-113.

Haussler, D., Krogh, A., Mian, I.S. and Sjölander, K. (1993) Protein Modeling using Hidden Markov Models: Analysis of Globins, Proceedings of the Hawaii International Conference on System Sciences, **1**, IEEE Computer Society Press, Los Alamitos, CA, 792-802.

Krogh, A., Brown, M., Mian, I. S., Sjölander, K. and Haussler, D. (1994a) Hidden Markov Models in Computational Biology: Applications to Protein Modeling. Journal of Molecular Biology, **235**, 1501–1531.

Krogh, A., Mian, I. S. and Haussler, D. (1994b) A Hidden Markov Model that Finds Genes in E. Coli DNA, Technical Report UCSC-CRL-93-33, University of California at Santa Cruz.

Lapedes, A., Barnes, C., Burks, C., Farber, R. and Sirotkin, K. Application of Neural Networks and Other Machine Learning Algorithms to DNA Sequence Analysis. In G.I. Bell and T.G. Marr, editors. The Procceedings of the Interface Between Computation Science and Nucleic Acid Sequencing Workshop. Proceedings of the Santa Fe Institute, volume VII, pages 157–182. Addison Wesley, Redwood City, CA, 1988.

Senapathy, P., Shapiro, M.B., and Harris, N.L. (1990) Splice Junctions, Branch Point Sites, and Exons: Sequence Statistics, Identification and Applications to Genome Project. Patterns in Nucleic Acid Sequences, Academic Press, 252–278.

Snyder, E.E. and Stormo, G.D. (1993) Identification of coding regions in genomic DNA sequences: an application of dynamic programming and neural networks. Nucleic Acids Research, **21**, 607–613.
